# An Object-Oriented Framework for the Simulation of Neural Nets

A. Linden    Th. Sudbrak    Ch. Tietz    F. Weber

German National Research Center for Computer Science

D-5205 Sankt Augustin 1, Germany

## Abstract

The field of software simulators for neural networks has been expanding very rapidly in the last years but their importance is still being underestimated. They must provide increasing levels of assistance for the design, simulation and analysis of neural networks. With our object-oriented framework (SESAME) we intend to show that very high degrees of transparency, manageability and flexibility for complex experiments can be obtained. SESAME's basic design philosophy is inspired by the natural way in which researchers explain their computational models. Experiments are performed with networks of building blocks, which can be extended very easily. Mechanisms have been integrated to facilitate the construction and analysis of very complex architectures. Among these mechanisms are the automatic configuration of building blocks for an experiment and multiple inheritance at run-time.

## 1  Introduction

In recent years a lot of work has been put into the development of simulation systems for neural networks [1, 2, 3, 4, 5, 6, 7, 9, 10, 11, 12]. Unfortunately their importance has been largely underestimated. In future, software environments will provide increasing levels of assistance for the design, simulation and analysis of neural networks as well as for other pattern and signal processing architectures. Yet large improvements are still necessary in order to fulfill the growing demands of the research community. Despite the existence of at least 100 software simulators, only very few of them can deal with, e. g. multiple learning paradigms and applications,

very large experiments.

In this paper we describe an object oriented framework for the simulation of neural networks and try to illustrate its flexibility, transparency and extendability. The prototype called SESAME has been implemented using C++ (on UNIX workstations running X-Windows) and currently consists of about 39.000 lines of code, implementing over 80 classes for neural network algorithms, pattern handling, graphical output and other utilities.

## 2    Philosophy of Design

The main objective of SESAME is to allow for arbitrary combinations of different learning and pattern processing paradigms (e. g. supervised, unsupervised, self-supervised or reinforcement learning) and different application domains (e. g. pattern recognition, vision, speech or control). To some degree the design of SESAME has been based on the observation that many researchers explain their neural information processing systems (NIPS) with block-diagrams. Such a block diagram consists of a group of primitive elements (building blocks). Each building block has inputs and outputs and a functional relationship between them. Connections describe the flow of data between the building blocks. Scripts related to the building blocks specify the flow of control. Complex NIPS are constructed from a library of building blocks (possibly themselves whole NIPS), which are interconnected via uniform communication links.

## 3    SESAME Design and Features

All building blocks share a list of common components. They all have *insites* and *outsites* that build the endpoints of communication links. *Datafields* contain the data (e. g. weight matrices or activation vectors) which is sent over the links. *Action functions* process input from the insites, update the internal state and compute appropriate outputs, e. g. performing weight updates and propagating activation or error vectors. *Command functions* provide a uniform user interface for all building blocks. *Scripts* control the execution of action or command functions or other scripts. They may contain conditional statements and loops as control structures. Furthermore a *symbol table* allows run-time access to parameters of the building block as learning rates, sizes, data ranges etc. Many other internal data structures and routines are provided for the *administration* and maintainance of building blocks.

The description of an experiment may be divided into the functional description of the building blocks (which can be done either in C++ or in the high-level description language, see below), the connection topology of the building blocks used, the control flow defined by scripts and a set of parameters for each of the building blocks.

# Design Highlights

## 3.1   User Interface

The user interface is text oriented and may be used interactively as well as script driven. This implies that any command that the user may choose interactively can also be used in a command file that is called non-interactively. This allows the easy adaption of arbitrary user interface structures from a primitive batch interface for large offline simulations to a fancy graphical user interface for online experiments.

Another consequence is that experiments are specified in the same command language that is used for the user interface. The user may thus easily switch from description files from previously saved experiments to the interactive manipulation of already loaded ones. Since the complete structure of an experiment is accessible at runtime, this not only means manipulation of parameters but also includes any imaginable modification of the experiment topology. The experienced user can, for example, include new building blocks for experiment observation or statistical evaluation and connect them to any point of the communication structure. Deletion of building blocks is possible, as well as modifying control scripts. The complete state of the experiment (i. e. the current values of all relevant data) can be saved for later experiments.

## 3.2   Hierarchies

In SESAME we distinguish two kinds of building blocks: *terminal* and *non-terminal* blocks. Non-terminal building blocks are used to structure a complex experiment into hierarchies of abstract building blocks containing substructures of an experiment that may themselves contain hierarchies of substructures. Terminal building blocks provide the data structures and primitive functions that are used in scripts (of non-terminal blocks) to compose the network algorithms. A non-terminal building block hides its internal structure and provides abstract sites and scripts as an interface to its internals. Therefore it appears as a terminal building block to the outside and may be used as such for the construction of an experiment. This construction is equivalent to the building of a single non-terminal building block (the *Experiment*) that encloses the complete experiment structure.

## 3.3   Construction of New Building Blocks

The functionality of SESAME can be extended using two different approaches. New terminal building blocks can be programmed deriving from existing C++ classes or new non-terminal building blocks may be assembled by using previously defined building blocks:

### 3.3.1   Programming New Terminal Building Blocks

Terminal building blocks can be designed by derivation from already existing C++ classes. The complete administration structure and possible predefined properties are inherited from the parent classes. In order to add new properties — e. g. new action functions, symbols, datafields, insites or outsites — a set of basic operations is being provided by the framework. One should note that new algorithms

and structures can be added to a class without any changes to the framework of SESAME.

### 3.3.2   Composing New Non-Terminal Building Blocks

Non-terminal building blocks can be combined from libraries of already designed terminal or non-terminal blocks. See for an example fig. ??, where a set of building blocks build a multilayer net which can be collapsed into one building block and reused in other contexts. Here insites and outsites define an interface between building blocks on adjacent levels of the experiment hierarchy. The flow of data inside the new building block is controlled by scripts that call action functions or scripts of its components. Such an abstract building block may be saved in a library for reuse. Even whole experiments can be collapsed to one building block leaving a lot of possibilities for the experimenter to cope with very large and complicated experiments.

### 3.3.3   Deriving New Non-Terminal Building Blocks

A powerful mechanism for organizing very complex experiments and allowing high degrees of flexibility and reuse is offered by the concept of inheritance. The basic mechanism executes the description of the parent building block and thereafter the description of the refinements for the derived block. All this may be done interactively, thus additional refinements can be added at runtime. Even the set of formal parameters of a block may be inherited and/or refined. Multiple inheritance is also possible.

For an example consider a general function approximator which may be used at many points in a more complex architecture. It can be implemented as an abstract base building block, only supplying basic structure as input and output and basic operations as "propagate input" and "train". Derivations of it then implement the algorithm and structure actually used. Statistical routines, visualization facilities, pattern handling and other utilities can be added as further specializations to a basic function approximator.

### 3.3.4   Parameters and Generic Building Blocks

A building block may also define formal parameters that allow the user to configure it at the time of its instantiation or inclusion into some other non-terminal building block. Thus non-terminal building blocks can be generic. They may be parameterized with types for interior building blocks, names of scripts etc. With this mechanism a multilayer net can be created with an arbitrary type of node or weight layers.

### 3.4   Autoconfiguration

When a user defines an experiment, only parameters that are really important must be specified. Redundant parameters, that depend on other paremeters of other building blocks, can often be determined automatically. In SESAME this is done via a constraint satisfaction process. Not only does this mechanism avoid specification of redundant information and check experiment parameters for consistency, but it

also enables the construction of generic structures. Communication links between outsites and insites of building blocks check data for matching types. Building blocks impose additional constraints on the data formats of their own sites. Constraints are formed upon information about the base types, dimensions, sizes and ranges of the data sent between the sites. The primary source of information are the parameters given to the building blocks at the time of their instantiation. After building the whole experiment, a propagation mechanism iteratively tries to complete missing information in order to satisfy all constraints. Thus information which is determined in one building block of the experiment may spread all over the experiment topology. As an example one can think of a building block which loads patterns from a file. The dimensionality of these patterns may be used automatically to configure building blocks holding weight layers for a multilayer network.

This autoconfiguration can be considered as finding the unique solution of set of equations where three cases may occur: inconsistency (contradiction between two information sources at one site), deadlock (insufficient information for a site) or success (unique solution). Inconsistencies are a proof of an erroneous design. Deadlocks indicate that the user has missed something.

### 3.5   Experiment Observation

Graphical output, file I/O or statistical analysis are usually not performed within the normal building blocks which comprise the network algorithms. These features are built into specialized utility building blocks that can be integrated at any point of the experiment topology, even during experiment runs.

## 4   Classes of Building Blocks

SESAME supports a rich taxonomy of building blocks for experiment construction:

For neural networks one can use building blocks for complete *node* and *weight* layers to construct multilayer networks. This granulation was chosen to allow for a more efficient way of computation than with building blocks that contain single neurons only. This level of abstraction still captures enough flexibility for many paradigms of NIPS. However, terminal building blocks for complete classes of neural nets are also provided if efficiency is first demand.

*Mathematical* building blocks perform arithmetic, trigonometric or more general mathematical transformations, as scaling and normalization. Building blocks for *coding* provide functionality to encode or decode patterns.

*Utility* building blocks provide access to the filesystem, where not only input or output files can be dealt with but also other UNIX processes by means of pipes. Others simply store structured or unstructured patterns to make them randomly accessible.

*Graphical* building blocks can be used to display any kind of data no matter if weight matrices, activation or error vectors are involved. This is a consequence of the abstract view of combining building blocks with different functionality but a uniform data interface. There are special building blocks for *analysis* which allow for clustering, averaging, error analysis, plotting and other statistical evaluations.

Finally simulations (cartpole, robot-arms etc.) can also be incorporated into building blocks. Real-world applications or other software packages can be accessed via specialized interface blocks.

## 5  Examples

Some illustrative examples for experiments can be found in [?] and many additional and more complex examples in the SESAME documentation. The full documentation as well as the software are available via ftp (see below).

Here we sketch only briefly, how paradigms and applications from different domains can be easily glued together as a natural consequence of the design of SESAME. Figure ?? shows part of an experiment in which a robot arm is controlled via a modified Kohonen feature map and a potential field path planner. The three building blocks, *workspace*, *map* and *planner* form the main part of the experiment. *Workspace* contains the simulation for the controlled robot arm and its graphical display and *map* contains the feature map that is used to transform the map coordinates proposed by *planner* to robot arm configurations. The map has been trained in another experiment to map the configuration space of the robot arm and the planner may have stored the positions of obstacles with respect to the map coordinates in still another experiment. The configuration and obstacle map have been saved as the results of the earlier experiments and are reused here. The *map* was taken from a library that contains different flavors of feature maps in form of nonterminal building blocks and hides the details of its complicated inner structure. The *Views* help to visualize the experiment and the *Buffers* are used to provide start values for the experiment runs. A *Subtractor* is shown that generates control inputs for the *workspace* by simply performing vector subtraction on subsequently proposed state vectors for the robot arm simulation.

## 6  Epilogue

We designed an object-oriented neural network simulator to cope with the increasing demands imposed by the current lines of research. Our implementation offers a high degree of flexibility for the experimental setup. Building blocks may be combined to build complex experiments in short development cycles. The simulator framework provides mechanisms to detect errors in the experiment setup and to provide parameters for generic subexperiments. A prototype was built, that is in use as our main research tool for neural network experiments and is constantly refined. Future developments are still necessary, e. g. to provide a graphical interface and more elegant mechanisms for the reuse of predefined building blocks. Further research issues are the parallelization of SESAME and the compilation of experiment parts to optimize their performance.

The software and its preliminary documentation can be obtained via ftp at **ftp.gmd.de** in the directory **gmd/as/sesame**. Unfortunately we cannot provide professional support at this moment.

Acknowledgments go to the numerous programmers and users of SESAME for all the work, valuable discussions and hints.

# References

[1] B. Angeniol and P. Treleaven. The PYGMALION neural network programming environment. In R. Eckmiller, editor, *Advanced Neural Computers*, pages 167 – 175, Amsterdam, 1990. Elsevier Science Publishers B. V. (North-Holland).

[2] N. Goddard, K. Lynne, T. Mintz, and L. Bukys. Rochester connectionist simulator. Technical Report TR-233 (revised), Computer Science Dept, University of Rochester, 1989.

[3] G. L. Heileman, H. K. Brown, and Georgiopoulos. Simulation of artificial neural network models using an object-oriented software paradigm. In *Proceedings of the International Joint Conference on Neural Networks*, pages II–133 – II–136, Washington, DC, 1990.

[4] NeuralWare Inc. Neuralworks professional ii user manual. 1989.

[5] T. T. Kraft. ANSpec tutorial workbook. San Diego, CA, 1990.

[6] T. Lange, J.B. Hodges, M. Fuenmayor, and L. Belyaev. Descartes: Development environment for simulating hybrid connectionist architectures. In *Proceedings of the Eleventh Annual Conference of the Cognitive Science Society, Ann Arbor, MI, August 1989*, 1989.

[7] A. Linden and C. Tietz. Combining multiple neural network paradigms and applications using SESAME. In *Proceedings of the Internation Joint Conference on Neural Networks IJCNN – Baltimore*. IEEE, 1992.

[8] Y. Miyata. A user's guide to SunNet version 5.6 – a tool for constructing, running, and looking into a PDP network. 1990.

[9] J. M. J. Murre and S. E. Kleynenberg. The MetaNet network environment for the development of modular neural networks. In *Proceedings of the International Neural Network Conference, Paris, 1990*, pages 717 – 720. IEEE, 1990.

[10] M.A. Wilson, S.B. Upinder, J.D. Uhley, and J.M. Bower. GENESIS: A system for simulating neural networks. In David S. Touretzky, editor, *Advances in Neural Information Processing Systems I*, pages 485–492. Morgan Kaufmann, 1988. Collected papers of the IEEE Conference on Neural Information Processing Systems – Natural and Synthetic, Denver, CO, November 1988.

[11] A. Zell, N. Mache, T. Sommer, and T. Korb. Recent developments of the snns neural network simulator. In *SPIE Conference on Applications of Artificial Neural Networks*. Universit"at Stuttgart, April 1991.

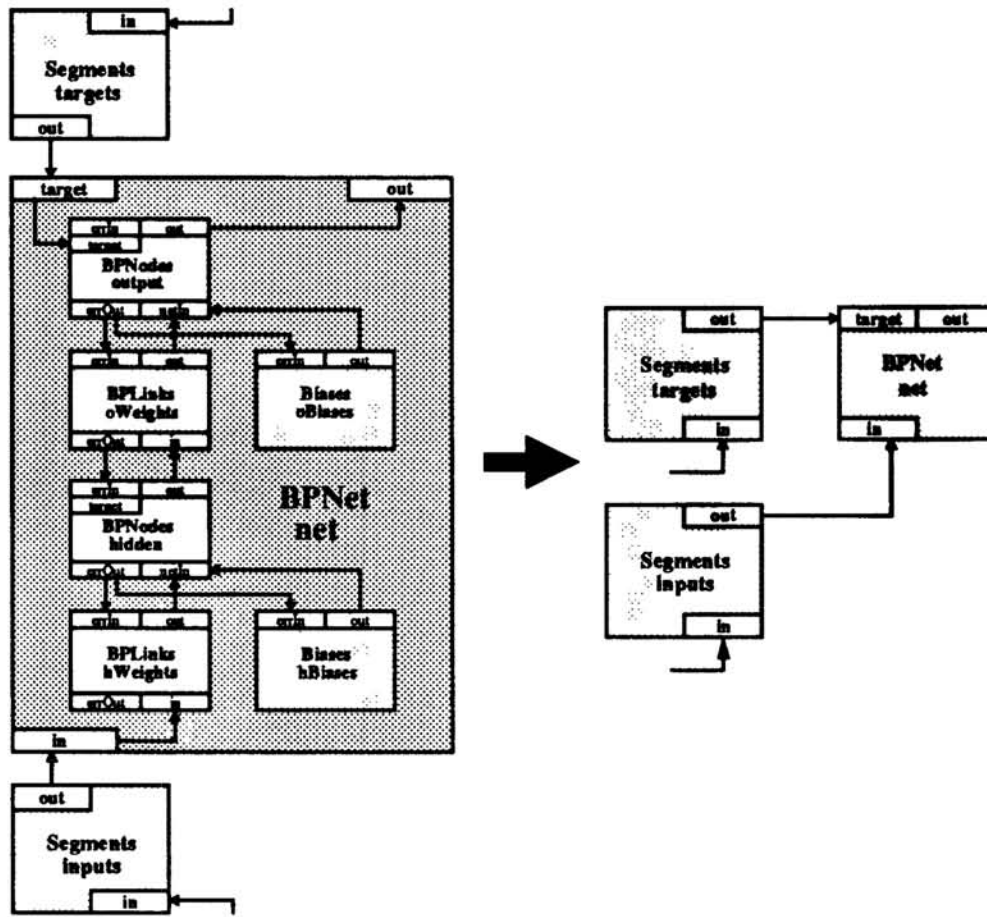

Figure 1: Integration of several terminal building blocks into a non-terminal building block with the Backpropagation example.

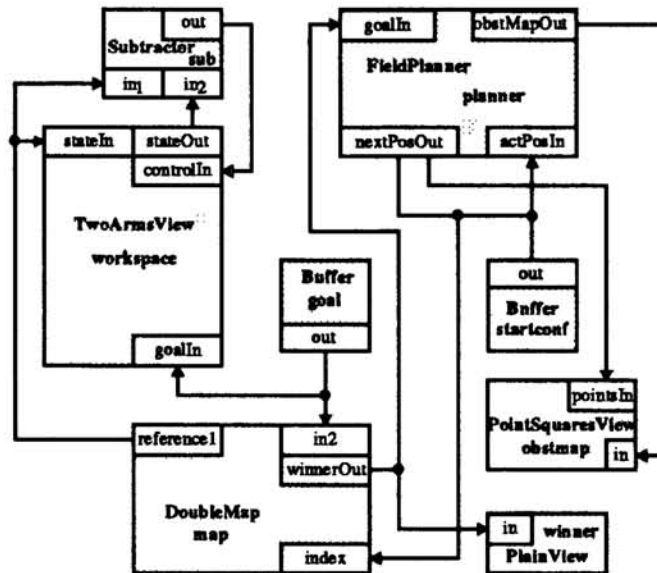

Figure 2: Robot arm control with a hybrid controller